# Bias-Optimal Incremental Problem Solving

**Jürgen Schmidhuber**
IDSIA, Galleria 2, 6928 Manno-Lugano, Switzerland
`juergen@idsia.ch`

## Abstract

Given is a problem sequence and a probability distribution (the *bias*) on programs computing solution candidates. We present an optimally fast way of incrementally solving each task in the sequence. Bias shifts are computed by program prefixes that modify the distribution on their suffixes by reusing successful code for previous tasks (stored in non-modifiable memory). No tested program gets more runtime than its probability times the total search time. In illustrative experiments, ours becomes the first general system to *learn* a universal solver for arbitrary $n$ disk *Towers of Hanoi* tasks (minimal solution size $2^n - 1$). It demonstrates the advantages of incremental learning by profiting from previously solved, simpler tasks involving samples of a simple context free language.

## 1 Brief Introduction to Optimal Universal Search

Consider an asymptotically optimal method for tasks with quickly verifiable solutions:

**Method 1.1** (LSEARCH) *View the $n$-th binary string $(0, 1, 00, 01, 10, 11, 000, \ldots)$ as a potential program for a universal Turing machine. Given some problem, for all $n$ do: every $2^n$ steps on average execute (if possible) one instruction of the $n$-th program candidate, until one of the programs has computed a solution.*

Given some problem class, if some unknown optimal program $p$ requires $f(k)$ steps to solve a problem instance of size $k$, and $p$ happens to be the $m$-th program in the alphabetical list, then LSEARCH (for *Levin Search*) [6] will need at most $O(2^m f(k)) = O(f(k))$ steps — the constant factor $2^m$ may be huge but does not depend on $k$. Compare [11, 7, 3].

Recently Hutter developed a more complex asymptotically optimal search algorithm for *all* well-defined problems [3]. HSEARCH (for *Hutter Search*) cleverly allocates part of the total search time for searching the space of proofs to find provably correct candidate programs with provable upper runtime bounds, and at any given time focuses resources on those programs with the currently best proven time bounds. Unexpectedly, HSEARCH manages to reduce the constant slowdown factor to a value of $1 + \epsilon$, where $\epsilon$ is an arbitrary positive constant. Unfortunately, however, the search in proof space introduces an unknown *additive* problem class-specific constant slowdown, which again may be huge.

In the real world, constants do matter. In this paper we will use basic concepts of optimal search to construct an optimal **incremental** problem solver that at any given time may exploit experience collected in previous searches for solutions to earlier tasks, to minimize the constants ignored by **non**incremental HSEARCH and LSEARCH.

## 2 Optimal Ordered Problem Solver (OOPS)

**Notation.** Unless stated otherwise or obvious, to simplify notation, throughout the paper newly introduced variables are assumed to be integer-valued and to cover the range clear from the context. Given some finite or infinite countable alphabet $Q = \{Q_1, Q_2, \ldots\}$, let $Q^*$ denote the set of finite sequences or strings over $Q$, where $\lambda$ is the empty string. We use the alphabet name's lower case variant to introduce (possibly variable) strings such as $q, q^1, q^2, \ldots \in Q^*$; $l(q)$ denotes the number of symbols in string $q$, where $l(\lambda) = 0$; $q_n$ is the $n$-th symbol of $q$; $q_{m:n} = \lambda$ if $m > n$ and $q_m q_{m+1} \ldots q_n$ otherwise (where $q_0 := q_{0:0} := \lambda$). $q^1 q^2$ is the concatenation of $q^1$ and $q^2$ (e.g., if $q^1 = abc$ and $q^2 = dac$ then $q^1 q^2 = abcdac$).

Consider countable alphabets $S$ and $Q$. Strings $s, s^1, s^2, \ldots \in S^*$ represent possible internal *states* of a computer; strings $q, q^1, q^2, \ldots \in Q^*$ represent code or programs for manipulating states. We focus on $S$ being the set of integers and $Q := \{1, 2, \ldots, n_Q\}$ representing a set of $n_Q$ instructions of some programming language (that is, substrings within states may also encode programs).

$R$ is a set of currently unsolved tasks. Let the variable $s(r) \in S^*$ denote the current state of task $r \in R$, with $i$-th component $s_i(r)$ on a *computation tape* $r$ (think of a separate tape for each task). For convenience we combine current state $s(r)$ and current code $q$ in a single *address space*, introducing negative and positive addresses ranging from $-l(s(r))$ to $l(q) + 1$, defining the content of address $i$ as $z(i)(r) := q_i$ if $0 < i \le l(q)$ and $z(i)(r) := s_{-i}(r)$ if $-l(s(r)) \le i \le 0$. All dynamic task-specific data will be represented at non-positive addresses. In particular, the current instruction pointer $ip(r) := z(a_{ip}(r))(r)$ of task $r$ can be found at (possibly variable) address $a_{ip}(r) \le 0$. Furthermore, $s(r)$ also encodes a modifiable probability distribution $p(r) = \{p_1(r), p_2(r), \ldots, p_{n_Q}(r)\}$ ($\sum_i p_i(r) = 1$) on $Q$. This variable distribution will be used to select a new instruction in case $ip(r)$ points to the current topmost address right after the end of the current code $q$.

$a_{frozen} \ge 0$ is a variable address that cannot decrease. Once chosen, the *code bias* $q_{0:a_{frozen}}$ will remain unchangeable forever — it is a (possibly empty) sequence of programs $q^1 q^2 \ldots$, some of them prewired by the user, others *frozen* after previous successful searches for solutions to previous tasks. Given $R$, the goal is to solve all tasks $r \in R$, by a program that appropriately uses or extends the current code $q_{0:a_{frozen}}$.

We will do this in a *bias-optimal* fashion, that is, no solution candidate will get much more search time than it deserves, given some initial probabilistic bias on program space $Q^*$:

**Definition 2.1** (BIAS-OPTIMAL SEARCHERS) *Given is a problem class $\mathcal{R}$, a search space $\mathcal{C}$ of solution candidates (where any problem $r \in \mathcal{R}$ should have a solution in $\mathcal{C}$), a task-dependent bias in form of conditional probability distributions $P(q \mid r)$ on the candidates $q \in \mathcal{C}$, and a predefined procedure that creates and tests any given $q$ on any $r \in \mathcal{R}$ within time $t(q, r)$ (typically unknown in advance). A searcher is $n$-bias-optimal ($n \ge 1$) if for any maximal total search time $T_{max} > 0$ it is guaranteed to solve any problem $r \in \mathcal{R}$ if it has a solution $p \in \mathcal{C}$ satisfying $t(p, r) \le P(p \mid r) T_{max}/n$.*

Unlike reinforcement learners [4] and heuristics such as *Genetic Programming* [2], OOPS (section 2.2) will be $n$-*bias-optimal*, where $n$ is a small and acceptable number, such as 8.

### 2.1 OOPS Prerequisites: Multitasking & Prefix Tracking Through Method "Try"

The Turing machine-based setups for HSEARCH and LSEARCH assume potentially infinite storage. Hence they may largely ignore questions of storage management. In any practical system, however, we have to efficiently reuse limited storage. This, and *multitasking,* is what the present subsection is about. The recursive method **Try** below allocates time to

program prefixes, each being tested on *multiple* tasks simultaneously, such that the *sum* of the runtimes of any given prefix, tested on all tasks, does not exceed the total search time multiplied by the prefix probability (the product of the tape-dependent probabilities of its previously selected components in $Q$). **Try** tracks effects of tested program prefixes, such as storage modifications (including probability changes) and partially solved task sets, to reset conditions for subsequent tests of alternative prefix continuations in an optimally efficient fashion (at most as expensive as the prefix tests themselves). Optimal backtracking requires that any prolongation of some prefix by some token gets immediately executed. To allow for efficient undoing of state changes, we use global Boolean variables $mark_i(r)$ (initially FALSE) for all possible state components $s_i(r)$. We initialize time $t_0 := 0$; probability $P := 1$; *q-pointer* $qp := a_{frozen}$ and state $s(r)$ (including $ip(r)$ and $p(r)$) with task-specific information for all task names $r$ in a *ring* $R_0$ of tasks. Here the expression *"ring"* indicates that the tasks are ordered in cyclic fashion; $\mid R \mid$ denotes the number of tasks in ring $R$. Given a global search time limit $T$, we **Try** to solve all tasks in $R_0$, by using existing code in $q = q_{1:qp}$ and / or by discovering an appropriate prolongation of $q$:

**Method 2.1** (BOOLEAN **Try** $(qp, r_0, R_0, t_0, P)$) *(returns* TRUE *or* FALSE; $r_0 \in R_0$).

**1.** *Make an empty stack $\mathcal{S}$; set local variables $r := r_0$; $R := R_0$; $t := t_0$;* Done:= FALSE.

WHILE $\mid R \mid > 0$ *and $t \leq PT$ and instruction pointer valid $(-l(s(r)) \leq ip(r) \leq qp)$ and instruction valid $(1 \leq z(ip(r))(r) \leq n_Q)$ and no halt condition (e.g., error such as division by 0) encountered* (evaluate conditions in this order until first satisfied, if any) DO:

*If possible, interpret / execute token $z(ip(r))(r)$ according to the rules of the given programming language (this may modify $s(r)$ including instruction pointer $ip(r)$ and distribution $p(r)$, but not $q$), continually increasing $t$ by the consumed time. Whenever the execution changes some state component $s_i(r)$ whose $mark_i(r) = $ FALSE, set $mark_i(r) :=$ TRUE and save the previous value $\hat{s}_i(r)$ by pushing the triple $(i, r, \hat{s}_i(r))$ onto $\mathcal{S}$. Remove $r$ from $R$ if solved.* IF $\mid R \mid > 0$, *set $r$ equal to the next task in ring $R$.* ELSE *set* Done := TRUE; $a_{frozen} := qp$ *(all tasks solved; new code frozen, if any).*

**2.** *Use $\mathcal{S}$ to efficiently reset only the modified $mark_i(k)$ to* FALSE *(but do not pop $\mathcal{S}$ yet).*

**3.** IF $ip(r) = qp + 1$ (**this means an online request for prolongation of the current prefix through a new token**): WHILE Done = FALSE *and there is some yet untested token $Z \in Q$ (untried since $t_0$ as value for $q_{qp+1}$), set $q_{qp+1} := Z$ and* Done := **Try** $(qp + 1, r, R, t, P * p(r)(Z))$, *where $p(r)(Z)$ is $Z$'s probability according to current $p(r)$.*

**4.** *Use $\mathcal{S}$ to efficiently restore only those $s_i(k)$ changed since $t_0$, thus also restoring instruction pointer $ip(r_0)$ and original search distribution $p(r_0)$. Return the value of* Done.

It is important that instructions whose runtimes are not known in advance can be interrupted by **Try** at any time. Essentially, **Try** conducts a depth-first search in program space, where the branches of the search tree are program prefixes, and backtracking is triggered once the sum of the runtimes of the current prefix on all current tasks exceeds the prefix probability multiplied by the total time limit. A successful **Try** will solve all tasks, possibly increasing $a_{frozen}$. In any case **Try** will completely restore all states of all tasks. Tracking / undoing effects of prefixes essentially does not cost more than their execution. So the $n$ in Def. 2.1 of *n-bias-optimality* is not greatly affected by backtracking: ignoring hardware-specific overhead, we lose at most a factor 2. An efficient iterative (non-recursive) version of **Try** for a broad variety of initial programming languages was implemented in C.

## 2.2 OOPS For Finding Universal Solvers

Now suppose there is an ordered sequence of tasks $r_1, r_2, \ldots$. Task $r_j$ may or may not depend on solutions for $r_i$ ($i, j = 1, 2, \ldots, j > i$). For instance, task $r_j$ may be to find a

faster way through a maze than the one found during the search for a solution to task $r_{j-1}$.

We are searching for a single program solving all tasks encountered so far (see [9] for variants of this setup). Inductively suppose we have solved the first $n$ tasks through programs stored below address $a_{frozen}$, and that the most recently found program starting at address $a_{last} \leq a_{frozen}$ actually solves all of them, possibly using information conveyed by earlier programs. To find a program solving the first $n + 1$ tasks, OOPS invokes **Try** as follows (using set notation for ring $R$):

**Method 2.2** (OOPS **(n+1)**) *Initialize* $T := 2$; $qp := a_{frozen}$.

**1.** *Set* $R = \{r_{n+1}\}$ *and* $ip(r_{n+1}) := a_{last}$. IF **Try** $(qp, r_{n+1}, R, 0, 0.5)$ *then exit.*

**2.** IF $n + 1 > T$ *go to* **3**. *Set* $R = \{r_1, r_2, \ldots, r_{n+1}\}$; *set local variable* $a := a_{frozen} + 1$; *for all* $r \in R$ *set* $ip(r) := a$. IF **Try** $(qp, r_{n+1}, R, 0, 0.5)$ *set* $a_{last} := a$ *and exit.*

**3.** *Set* $T := 2T$, *and go to* **1**.

That is, we spend roughly equal time on two simultaneous searches. The second (step **2**) considers all tasks and all prefixes. The first (step **1**), however, focuses only on task $n + 1$ and the most recent prefix and its possible continuations. In particular, start address $a_{last}$ does not increase as long as new tasks can be solved by prolonging $q_{a_{last}:a_{frozen}}$. Why is this justified? A bit of thought shows that it is impossible for the most recent code starting at $a_{last}$ to request any additional tokens that could harm its performance on previous tasks. We already inductively know that all of its prolongations will solve all tasks up to $n$.

Therefore, given tasks $r_1, r_2, \ldots$, we first initialize $a_{last}$; then for $i := 1, 2, \ldots$ invoke OOPS$(i)$ to find programs starting at (possibly increasing) address $a_{last}$, each solving all tasks so far, possibly eventually discovering a universal solver for all tasks in the sequence. As address $a_{last}$ increases for the $n$-th time, $q^n$ is defined as the program starting at $a_{last}$'s old value and ending right before its new value. Clearly, $q^m$ $(m > n)$ may exploit $q^n$.

**Optimality.** OOPS not only is asymptotically optimal in Levin's sense [6] (see Method 1.1), but also near bias-optimal (Def. 2.1). To see this, consider a program $p$ solving problem $r_j$ within $k$ steps, given current code bias $q_{0:a_{frozen}}$ and $a_{last}$. Denote $p$'s probability by $P(p)$. A bias-optimal solver would solve $r_j$ within at most $k/P(p)$ steps. We observe that OOPS will solve $r_j$ within at most $2^3 k/P(p)$ steps, ignoring overhead: a factor 2 might get lost for allocating half the search time to prolongations of the most recent code, another factor 2 for the incremental doubling of $T$ (necessary because we do not know in advance the best value of $T$), and another factor 2 for **Try**'s resets of states and tasks. So the method is *8-bias-optimal* (ignoring hardware-specific overhead) with respect to the current task.

Our only bias shifts are due to freezing programs once they have solved a problem. That is, unlike the learning rate-based bias shifts of ADAPTIVE LSEARCH [10], those of OOPS do not reduce probabilities of programs that were meaningful and executable *before* the addition of any new $q^i$. Only formerly meaningless, interrupted programs trying to access code for earlier solutions when there weren't any suddenly may become prolongable and successful, once some solutions to earlier tasks have been stored.

Hopefully we have $P(p) >> P(p')$, where $p'$ is among the most probable fast solvers of $r_j$ that do *not* use previously found code. For instance, $p$ may be rather short and likely because it uses information conveyed by earlier found programs stored below $a_{frozen}$. E.g., $p$ may call an earlier stored $q^i$ as a subprogram. Or maybe $p$ is a short and fast program that copies $q^i$ into state $s(r_j)$, then modifies the copy just a little bit to obtain $\bar{q}^i$, then successfully applies $\bar{q}^i$ to $r_j$. If $p'$ is not many times faster than $p$, then OOPS will in general suffer from a much smaller constant slowdown factor than LSEARCH, reflecting the extent to which solutions to successive tasks do share useful mutual information.

Unlike nonincremental LSEARCH and HSEARCH, which do **not** require online-generated programs for their aymptotic optimality properties, OOPS **does** depend on such programs: The currently tested prefix may temporarily rewrite the search procedure by invoking previously frozen code that redefines the probability distribution on its suffixes, based on experience ignored by LSEARCH & HSEARCH (**metasearching & metalearning!**).

As we are solving more and more tasks, thus collecting and freezing more and more $q^i$, it will generally become harder and harder to identify and address and copy-edit particular useful code segments within the earlier solutions. As a consequence we expect that much of the knowledge embodied by certain $q^j$ actually will be about how to access and edit and use programs $q^i$ $(i < j)$ previously stored below $q^j$.

## 3    A Particular Initial Programming Language

The efficient search and backtracking mechanism described in section 2.1 is not aware of the nature of the particular programming language given by $Q$, the set of initial instructions for modifying states. The language could be list-oriented such as LISP, or based on matrix operations for neural network-like parallel architectures, etc. For the experiments we wrote an interpreter for an exemplary, stack-based, universal programming language inspired by FORTH [8], whose disciples praise its beauty and the compactness of its programs.

Each task's tape holds its state: various stack-like data structures represented as sequences of integers, including a data stack $ds$ (with stack pointer $dp$) for function arguments, an auxiliary data stack $Ds$, a function stack $fns$ of entries describing (possibly recursive) functions defined by the system itself, a callstack $cs$ (with stack pointer $cp$ and top entry $cs[cp]$) for calling functions, where local variable $cs[cp].ip$ is the current *instruction pointer,* and *base pointer* $cs[cp].dp$ points into $ds$ below the values considered as arguments of the most recent function call: Any instruction of the form *inst $(x_1, \ldots, x_n)$* expects its $n$ arguments on top of $ds$, and replaces them by its return values. Illegal use of any instruction will cause the currently tested program prefix to halt. In particular, it is illegal to set variables (such as stack pointers or instruction pointers) to values outside their prewired ranges, or to pop empty stacks, or to divide by 0, or to call nonexistent functions, or to change probabilities of nonexistent tokens, etc. **Try** (Section 2.1) will interrupt prefixes as soon as their $t > TP$.

**Instructions.** We defined 68 instructions, such as *oldq(n)* for calling the $n$-th previously found program $q^n$, or *getq(n)* for making a copy of $q^n$ on stack $ds$ (e.g., to edit it with additional instructions). Lack of space prohibits to explain all instructions (see [9]) — we have to limit ourselves to the few appearing in solutions found in the experiments, using readable names instead of their numbers: Instruction *c1()* returns constant 1. Similarly for *c2(), ..., c5(). dec(x)* returns $x - 1$; *by2(x)* returns $2x$; *grt(x,y)* returns 1 if $x > y$, otherwise 0; *delD()* decrements stack pointer $Dp$ of $Ds$; *fromD()* returns the top of $Ds$; *toD()* pushes the top entry of $ds$ onto $Ds$; *cpn(n)* copies the n topmost $ds$ entries onto the top of $ds$, increasing $dp$ by $n$; *cpnb(n)* copies $n$ $ds$ entries above the $cs[cp].dp$-th $ds$ entry onto the top of $ds$; *exec(n)* interprets $n$ as the number of an instruction and executes it; *bsf(n)* considers the entries on stack $ds$ above its $cs[cp].dp + n$-th entry as code and uses callstack $cs$ to *call* this code (code is executed by step **1** of **Try** (Section 2.1), one instruction at a time; the instruction *ret()* causes a return to the address of the next instruction right after the calling instruction). Given $n$ input arguments on $ds$, instruction *defnp()* pushes onto $ds$ the begin of a definition of a procedure with $n$ inputs; this procedure returns if its topmost input is 0, otherwise decrements it. *callp()* pushes onto $ds$ code for a call of the most recently defined function / procedure. Both *defnp* and *callp* also push code for making a fresh copy of the inputs of the most recently defined code, expected on top of $ds$. *endnp()* pushes code for returning from the current call, then *calls* the code generated so far on stack $ds$ above the $n$ inputs, applying the code to a copy of the inputs on top of $ds$. *boostq(i)* sequentially goes through all tokens of the $i$-th self-discovered frozen

program, **boosting** each token's probability by adding $n_Q$ to its enumerator and also to the denominator shared by all instruction probabilities — denominator and all numerators are stored on tape, defining distribution $p(r)$.

**Initialization.** Given any task, we add task-specific instructions. We start with a maximum entropy distribution on the $> 68$ $Q_i$ (all numerators set to 1), then insert substantial prior bias by assigning the lowest (easily computable) instruction numbers to the task-specific instructions, and by **boosting** (see above) the initial probabilities of appropriate *"small number pushers"* (such as *c1, c2, c3*) that push onto *ds* the numbers of the task-specific instructions, such that they become executable as part of code on *ds*. We also boost the probabilities of the simple arithmetic instructions *by2* and *dec*, such that the system can easily create other integers from the probable ones (e.g., code sequence *(c3 by2 by2 dec)* will return integer 11). Finally we also boost *boostq*.

## 4   Experiments: Towers of Hanoi and Context-Free Symmetry

Given are $n$ disks of $n$ different sizes, stacked in decreasing size on the first of three pegs. Moving some peg's top disk to the top of another (possibly empty) peg, one disk at a time, but never a larger disk onto a smaller, transfer all disks to the third peg. Remarkably, the fastest way of solving this famous problem requires $2^n - 1$ moves ($n \geq 0$).

Untrained humans find it hard to solve instances $n > 6$. Anderson [1] applied traditional reinforcement learning methods and was able to solve instances up to $n = 3$, solvable within at most 7 moves. Langley [5] used learning production systems and was able to solve Hanoi instances up to $n = 5$, solvable within at most 31 moves. Traditional **non**learning planning procedures systematically explore all possible move combinations. They also fail to solve Hanoi problem instances with $n > 15$, due to the exploding search space (Jana Koehler, IBM Research, personal communication, 2002). OOPS, however, is searching in program space instead of raw solution space. Therefore, in principle it should be able to solve arbitrary instances by discovering the problem's elegant recursive solution: given $n$ and three pegs $S, A, D$ (source peg, auxiliary peg, destination peg), define procedure

**Method 4.1** (HANOI(**S,A,D,n**)) IF $n = 0$ *exit. Call* HANOI*(S, D, A, n-1); move top disk from S to D; call* HANOI*(A, S, D, n-1).*

The $n$-th task is to solve all Hanoi instances up to instance $n$. We represent the dynamic environment for task $n$ on the $n$-th task tape, allocating $n+1$ addresses for each peg, to store its current disk positions and a pointer to its top disk (0 if there isn't any). We represent pegs $S, A, D$ by numbers 1, 2, 3, respectively. That is, given an instance of size $n$, we push onto *ds* the values $1, 2, 3, n$. By doing so *we insert substantial, nontrivial prior knowledge* about problem size and the fact that it is useful to represent each peg by a symbol.

We add three instructions to the 68 instructions of our FORTH-like programming language: *mvdsk()* assumes that $S, A, D$ are represented by the first three elements on *ds* above the current base pointer $cs[cp].dp$, and moves a disk from peg $S$ to peg $D$. Instruction *xSA()* exchanges the representations of $S$ and $A$, *xAD()* those of $A$ and $D$ (combinations may create arbitrary peg patterns). Illegal moves cause the current program prefix to halt. Overall success is easily verifiable since our objective is achieved once the first two pegs are empty.

Within reasonable time (a week) on an off-the-shelf personal computer (1.5 GHz) the system was not able to solve instances involving more than 3 disks. This gives us a welcome opportunity to demonstrate its **incremental** learning abilities: we first trained it on an additional, easier task, to teach it something about recursion, hoping that this would help to solve the Hanoi problem as well. For this purpose we used a seemingly unrelated **symmetry problem** based on the context free language $\{1^n 2^n\}$: given input $n$ on the data stack *ds*, the goal is to place symbols on the auxiliary stack $Ds$ such that the $2n$ topmost elements

are $n$ 1's followed by $n$ 2's. We add two more instructions to the initial programming language: instruction *1toD()* pushes 1 onto *Ds*, instruction *2toD()* pushes 2. Now we have a total of five task-specific instructions (including those for Hanoi), with instruction numbers 1, 2, 3, 4, 5, for *1toD*, *2toD*, *mvdsk*, *xSA*, *xAD*, respectively.

So we first boost (Section 3) instructions *c1, c2* for the first training phase where the $n$-th task $(n = 1, \ldots, 30)$ is to solve all symmetry problem instances up to $n$. Then we undo the symmetry-specific boosts of *c1, c2* and boost instead the Hanoi-specific "instruction number pushers" $c3, c4, c5$ for the subsequent training phase where the $n$-th task (again $n = 1, \ldots, 30$) is to solve all Hanoi instances up to $n$.

**Results.** Within roughly 0.3 days, OOPS found and froze code solving the symmetry problem. Within 2 more days it also found a universal Hanoi solver, exploiting the benefits of incremental learning ignored by **non**incremental HSEARCH and LSEARCH. It is instructive to study the sequence of intermediate solutions. In what follows we will transform integer sequences discovered by OOPS back into readable programs (to fully understand them, however, one needs to know all side effects, and which instruction has got which number).

For the symmetry problem, within less than a second, OOPS found silly but working code for $n = 1$. Within less than 1 hour it had solved the 2nd, 3rd, 4th, and 5th instances, always simply prolonging the previous code without changing the start address $a_{last}$. The code found so far was unelegant: *(defnp 2toD grt c2 c2 endnp boostq delD delD bsf 2toD fromD delD delD delD fromD bsf by2 bsf by2 fromD delD delD fromD cpnb bsf)*. But it does solve all of the first 5 instances. Finally, after 0.3 days, OOPS had created and tested a new, elegant, recursive program (no prolongation of the previous one) with a new increased start address $a_{last}$, solving all instances up to 6: *(defnp c1 calltp c2 endnp)*. That is, it was cheaper to solve all instances up to 6 by discovering and applying this new program to all instances so far, than just prolonging old code on instance 6 only. In fact, the program turns out to be a universal symmetry problem solver. On the stack, it constructs a 1-argument procedure that returns nothing if its input argument is 0, otherwise calls the instruction *1toD* whose code is 1, then calls itself with a decremented input argument, then calls *2toD* whose code is 2, then returns. Using this program, within an additional 20 milliseconds, OOPS had also solved the remaining 24 symmetry tasks up to $n = 30$.

Then OOPS switched to the Hanoi problem. 1 *ms* later it had found trivial code for $n = 1$: *(mvdsk)*. After a day or so it had found fresh yet bizarre code (new start address $a_{last}$) for $n = 1, 2$: *(c4 c3 cpn c4 by2 c3 by2 exec)*. Finally, after 3 days it had found fresh code (new $a_{last}$) for $n = 1, 2, 3$: *(c3 dec boostq defnp c4 calltp c3 c5 calltp endnp)*. This already is an optimal universal Hanoi solver. Therefore, within 1 additional day OOPS was able to solve the remaining 27 tasks for $n$ up to 30, reusing the same program $q_{a_{last}:a_{frozen}}$ again and again. Recall that the optimal solution for $n = 30$ takes $> 10^9$ *mvdsk* operations, and that for each *mvdsk* several other instructions need to be executed as well!

The final Hanoi solution profits from the earlier recursive solution to the symmetry problem. How? The prefix *(c3 dec boostq)* (probability 0.003) temporarily rewrites the search procedure (this illustrates the benefits of **metasearching!**) by exploiting previous code: Instruction *c3* pushes 3; *dec* decrements this; *boostq* takes the result 2 as an argument and thus boosts the probabilities of all components of the 2nd frozen program, which happens to be the previously found universal symmetry solver. This leads to an online bias shift that greatly increases the probability that *defnp, calltp, endnp,* will appear in the suffix of the online-generated program. These instructions in turn are helpful for building (on the data stack *ds*) the double-recursive procedure generated by the suffix *(defnp c4 calltp c3 c5 calltp endnp)*, which essentially constructs a 4-argument procedure that returns nothing if its input argument is 0, otherwise decrements the top input argument, calls the instruction *xAD* whose code is 4, then calls itself, then calls *mvdsk* whose code is 5, then calls *xSA* whose code is 3, then calls itself again, then returns (compare the standard Hanoi solution).

The total probability of the final solution, given the previous codes, is $0.325 * 10^{-10}$. On the other hand, the probability of the essential Hanoi code *(defnp c4 calltp c3 c5 calltp endnp)*, given nothing, is only $4 * 10^{-14}$, which explains why it was not quickly found without the help of an easier task. So in this particular setup the **incremental** training due to the simple recursion for the symmetry problem indeed provided useful training for the more complex Hanoi recursion, speeding up the search by a factor of roughly 1000.

The entire 4 day search tested 93,994,568,009 prefixes corresponding to 345,450,362,522 instructions costing 678,634,413,962 time steps (some instructions cost more than 1 step, in particular, those making copies of strings with length $> 1$, or those increasing the probabilities of more than one instruction). **Search time of an optimal solver is a natural measure of initial bias.** Clearly, most tested prefixes are short — they either halt or get interrupted soon. Still, some programs do run for a long time; the longest measured runtime exceeded 30 billion steps. The stacks $\mathcal{S}$ of recursive invocations of **Try** for storage management (Section 2.1) collectively never held more than 20,000 elements though.

Different initial bias will yield different results. E.g., we could set to zero the initial probabilities of most of the 73 initial instructions (most are unnecessary for our two problem classes), and then solve all $2 \times 30$ tasks more quickly (at the expense of obtaining a non-universal initial programming language). The point of this experimental section, however, is not to find the most reasonable initial bias for particular problems, but to illustrate the general functionality of the first general near-bias-optimal incremental learner. In ongoing research we are equipping OOPS with neural network primitives and are applying it to robotics. Since OOPS will scale to larger problems in essentially unbeatable fashion, the hardware speed-up factor of $10^9$ expected for the next 30 years appears promising.

## References

[1] C. W. Anderson. *Learning and Problem Solving with Multilayer Connectionist Systems*. PhD thesis, University of Massachusetts, Dept. of Comp. and Inf. Sci., 1986.

[2] N. L. Cramer. A representation for the adaptive generation of simple sequential programs. In J.J. Grefenstette, editor, *Proceedings of an International Conference on Genetic Algorithms and Their Applications, Carnegie-Mellon University, July 24-26, 1985*, Hillsdale NJ, 1985. Lawrence Erlbaum Associates.

[3] M. Hutter. The fastest and shortest algorithm for all well-defined problems. *International Journal of Foundations of Computer Science*, 13(3):431–443, 2002.

[4] L.P. Kaelbling, M.L. Littman, and A.W. Moore. Reinforcement learning: a survey. *Journal of AI research*, 4:237–285, 1996.

[5] P. Langley. Learning to search: from weak methods to domain-specific heuristics. *Cognitive Science*, 9:217–260, 1985.

[6] L. A. Levin. Universal sequential search problems. *Problems of Information Transmission*, 9(3):265–266, 1973.

[7] M. Li and P. M. B. Vitányi. *An Introduction to Kolmogorov Complexity and its Applications (2nd edition)*. Springer, 1997.

[8] C. H. Moore and G. C. Leach. FORTH - a language for interactive computing, 1970. http://www.ultratechnology.com.

[9] J. Schmidhuber. Optimal ordered problem solver. Technical Report IDSIA-12-02, arXiv:cs.AI/0207097 v1, IDSIA, Manno-Lugano, Switzerland, July 2002.

[10] J. Schmidhuber, J. Zhao, and M. Wiering. Shifting inductive bias with success-story algorithm, adaptive Levin search, and incremental self-improvement. *Machine Learning*, 28:105–130, 1997.

[11] R.J. Solomonoff. An application of algorithmic probability to problems in artificial intelligence. In L. N. Kanal and J. F. Lemmer, editors, *Uncertainty in Artificial Intelligence*, pages 473–491. Elsevier Science Publishers, 1986.
